# Statistical Mechanics of Temporal Association in Neural Networks with Delayed Interactions

**Andreas V.M. Herz**
Division of Chemistry
Caltech 139-74
Pasadena, CA 91125

**Zhaoping Li**
School of Natural Sciences
Institute for Advanced Study
Princeton, NJ 08540

**J. Leo van Hemmen**
Physik-Department
der TU München
D-8046 Garching, FRG

## Abstract

We study the representation of static patterns and temporal associations in neural networks with a broad distribution of signal delays. For a certain class of such systems, a simple intuitive understanding of the spatio-temporal computation becomes possible with the help of a novel Lyapunov functional. It allows a quantitative study of the asymptotic network behavior through a statistical mechanical analysis. We present analytic calculations of both retrieval quality and storage capacity and compare them with simulation results.

## 1 INTRODUCTION

Basic computational functions of associative neural structures may be analytically studied within the framework of attractor neural networks where *static patterns* are stored as stable *fixed-points* for the system's dynamics. If the interactions between single neurons are instantaneous and mediated by symmetric couplings, there is a Lyapunov function for the retrieval dynamics (Hopfield 1982). The global computation corresponds in that case to a downhill motion in an energy landscape created by the stored information. Methods of equilibrium statistical mechanics may be applied and permit a quantitative analysis of the asymptotic network behavior (Amit et al. 1985, 1987). The existence of a Lyapunov function is thus of great conceptual as well as technical importance. Nevertheless, one should be aware that environmental inputs to a neural net always provide information in *both space and time*. It is therefore desirable to extend the original Hopfield scheme and to explore possibilities for a joint representation of static patterns and temporal associations.

Signal delays are omnipresent in the brain and play an important role in biological information processing. Their incorporation into theoretical models seems to be rather convincing, especially if one includes the distribution of the delay times involved. Kleinfeld (1986) and Sompolinsky and Kanter (1986) proposed models for temporal associations, but they only used a *single* delay line between two neurons. Tank and Hopfield (1987) presented a feedforward architecture for sequence recognition based on multiple delays, but they just considered information relative to the *very end* of a given sequence. Besides these deficiences, both approaches lack the quality to acquire knowledge through a true learning mechanism: Synaptic efficacies have to be calculated by hand which is certainly not satisfactory both from a neurobiological point of view and also for applications in artificial intelligence.

This drawback has been overcome by a careful interpretation of the Hebb principle (1949) for neural networks with a *broad* distribution of transmission delays (Herz et al. 1988, 1989). After the system has been taught stationary patterns and temporal sequences — by the *same* principle ! — it reproduces them with high precission when triggered suitably. In the present contribution, we focus on a special class of such delay networks and introduce a Lyapunov (energy) functional for the deterministic retrieval dynamics (Li and Herz 1990). We thus generalize Hopfield's approach to the domain of temporal associations. Through an extension of the usual formalism of equilibrium statistical mechanics to time-dependent phenomena, we analyze the network performance under a stochastic (noisy) dynamics. We derive quantitative results on both the retrieval quality and storage capacity, and close with some remarks on possible generalizations of this approach.

## 2 DYNAMICS OF THE NEURONS

Throughout what follows, we describe a neural network as a collection of $N$ two-state neurons with activities $S_i = 1$ for a firing cell and $S_i = -1$ for a quiescent one. The cells are connected by synapses with modifiable efficacies $J_{ij}(\tau)$. Here $\tau$ denotes the delay for the information transport from $j$ to $i$. We focus on a soliton-like propagation of neural signals, characteristic for the (axonal) transmission of action potentials, and consider a model where each pair of neurons is linked by *several* axons with delays $0 \leq \tau \leq \tau_{\max}$. Other architectures with only a single link have been considered elsewhere (Coolen and Gielen 1988; Herz et al. 1988, 1989; Kerszberg and Zippelius 1990). External stimuli are fed into the system via receptors $\sigma_i = \pm 1$ with input sensitivity $\gamma$. The postsynaptic potentials are given by

$$h_i(t) = (1-\gamma) \sum_{j=1}^{N} \sum_{\tau=0}^{\tau_{\max}} J_{ij}(\tau) S_j(t-\tau) + \gamma \sigma_i(t) . \tag{1}$$

We concentrate on synchronous dynamics (Little 1974) with basic time step $\Delta t = 1$. Consequently, signal delays take nonnegative integer values. Synaptic noise is described by a stochastic Glauber dynamics with noise level $\beta = T^{-1}$ (Peretto 1984),

$$\text{Prob}\left[S_i(t+1) = \pm 1\right] = \frac{1}{2}\left\{1 \pm \tanh\left[\beta h_i(t)\right]\right\} , \tag{2}$$

where Prob denotes probability. For $\beta \to \infty$, we arrive at a deterministic dynamics,

$$S_i(t+1) = \text{sgn}[h_i(t)] \equiv \begin{cases} 1, & \text{if } h_i(t) > 0 \\ -1, & \text{if } h_i(t) < 0 \end{cases} . \tag{3}$$

## 3 HEBBIAN LEARNING

During a learning session the synaptic strengths may change according to the Hebb principle (1949). We focus on a connection with delay $\tau$ between neurons $i$ and $j$. According to Hebb, the corresponding efficacy $J_{ij}(\tau)$ will be increased if cell $j$ takes part in *firing* cell $i$. In its physiological context, this rule was originaly formulated for excitatory synapses only, but for simplicity, we apply it to all synapses.

Due to the delay $\tau$ in (1) and the parallel dynamics (2), it takes $\tau+1$ time steps until neuron $j$ actually influences the *state* of neuron $i$. $J_{ij}(\tau)$ thus changes by an amount proportional to the product of $S_j(t-\tau)$ and $S_i(t+1)$. Starting with $J_{ij}(\tau)=0$, we obtain after $P$ learning sessions, labeled by $\mu$ and each of duration $D_\mu$,

$$J_{ij}(\tau) = \varepsilon(\tau)N^{-1}\sum_{\mu=1}^{P}\sum_{t_\mu=1}^{D_\mu} S_i(t_\mu+1)S_j(t_\mu-\tau) \equiv \varepsilon(\tau)\tilde{J}_{ij}(\tau) \ . \tag{4}$$

The parameters $\varepsilon(\tau)$, normalized by $\sum_{\tau=0}^{\tau_{max}}\varepsilon(\tau) = 1$, take morphological characteristics of the delay lines into account; $N^{-1}$ is a scaling factor useful for the theoretical analysis. By (4), synapses act as microscopic feature detectors during the learning sessions and store correlations of the taught sequences in both space $(i,j)$ and time $(\tau)$. In general, they will be asymmetric in the sense that $J_{ij}(\tau) \neq J_{ji}(\tau)$.

During learning, we set $T = 0$ and $\gamma = 1$ to achieve a "clamped learning scenario" where the system evolves strictly according to the external stimuli, $S_i(t_\mu)=\sigma_i(t_\mu-1)$. We study the case where all input sequences $\sigma_i(t_\mu)$ are cyclic with equal periods $D_\mu = D$, i.e., $\sigma_i(t_\mu) = \sigma_i(t_\mu \pm D)$ for all $\mu$. In passing we note that one should offer the sequences already $\tau_{max}$ time steps before allowing synaptic plasticity à la (4) so that both $S_i$ and $S_j$ are in well defined states during the actual learning sessions. We define patterns $\xi_{ia}^{\mu 0}$ by $\xi_{ia}^{\mu 0} \equiv \sigma_i(t_\mu = a)$ for $0 \leq a < D$ and get

$$J_{ij}(\tau) = \varepsilon(\tau)N^{-1}\sum_{\mu=1}^{P}\sum_{a=0}^{D-1} \xi_{i,a+1}^{\mu 0}\xi_{j,a-\tau}^{\mu 0} \ . \tag{5}$$

Our learning scheme is thus a generalization of outer-product rules to spatio-temporal patterns. As in the following, temporal arguments of the sequence pattern states $\xi$ and the synaptic couplings should always be understood *modulo D*.

## 4 LYAPUNOV FUNCTIONAL

Using formulae (1)-(5), one may derive equations of motion for macroscopic order parameters (Herz et al. 1988, 1989) but this kind of analysis only applies to the case $P \ll \log N$. However, note that from (4) and (5), we get $\tilde{J}_{ij}(\tau) = \tilde{J}_{ji}(D - (2+\tau))$. For all networks whose *a priori* weights $\varepsilon(\tau)$ obey $\varepsilon(\tau) = \varepsilon(D - (2 + \tau))$ we have thus found an "extended synaptic symmetry" (Li and Herz 1990),

$$J_{ij}(\tau) = J_{ji}(D - (2 + \tau)) \ , \tag{6}$$

generalizing Hopfield's symmetry assumption $J_{ij} = J_{ji}$ in a natural way to the temporal domain. To establish a Lyapunov functional for the noiseless retrieval

dynamics (3), we take $\gamma = 0$ in (1) and define

$$H(t) \equiv -\frac{1}{2} \sum_{i,j=1}^{N} \sum_{a,\tau=0}^{D-1} J_{ij}(\tau) S_i(t-a) S_j(t-(a+\tau+1)\%D) , \qquad (7)$$

where $a\%b \equiv a \bmod b$. The functional $H$ depends on *all* states between $t+1-D$ and $t$ so that solutions with *constant* $H$, like $D$-periodic cycles, need not be static fixed points of the dynamics. By (1), (5) and (6), the difference $\Delta H(t) \equiv H(t) - H(t-1)$ is

$$\Delta H(t) = -\sum_{i=1}^{N}[S_i(t)-S_i(t-D)]h_i(t-1) - \frac{\varepsilon(D-1)}{2N} \sum_{\mu=1}^{P} \sum_{a=0}^{D-1} \{\sum_{i=1}^{N} \xi_{ia}^{\mu 0}[S_i(t)-S_i(t-D)]\}^2 .$$

$$(8)$$

The dynamics (3) implies that the first term is nonpositive. Since $\varepsilon(\tau) \geq 0$, the same holds true for the second one. For finite $N$, $H$ is bounded and $\Delta H$ has to vanish as $t \to \infty$. The system therefore settles into a state with $S_i(t) = S_i(t-D)$ for all $i$.

We have thus exposed two important facts: (a) the retrieval dynamics is governed by a Lyapunov functional, and (b) the system relaxes to a static state or a limit cycle with $S_i(t) = S_i(t-D)$ — oscillatory solutions with the *same* period as that of the taught cycles or a period which is equal to an integer fraction of $D$.

Stepping back for an overview, we notice that $H$ is a Lyapunov functional for all networks which exhibit an "extended synaptic symmetry" (6) and for which the matrix $\mathbf{J}(D-1)$ is positive semi-definite. The Hebbian synapses (4) constitute an important special case and will be the main subject of our further discussion.

## 5 STATISTICAL MECHANICS

We now prove that a limit cycle of the retrieval dynamics indeed resembles a stored sequence. We proceed in two steps. First, we demonstrate that our task concerning cyclic temporal associations can be mapped onto a symmetric network without delays. Second, we apply equilibrium statistical mechanics to study such *"equivalent systems"* and derive analytic results for the retrieval quality and storage capacity.

$D$-periodic oscillatory solutions of the retrieval dynamics can be interpreted as static states in a "$D$-plicated" system with $D$ columns and $N$ rows of cells with activities $S_{ia}$. A network state will be written $A = (A_0, A_1, \ldots, A_{D-1})$ with $A_a \equiv \{S_{ia}; 1 \leq i \leq N\}$. To reproduce the parallel dynamics of the original system, neurons $S_{ia}$ with $a = t\%D$ are updated at time $t$. The time evolution of the new network therefore has a pseudo-sequential characteristic: synchronous within single columns and sequentially ordered with respect to these columns. Accordingly, the neural activities at time $t$ are given by $S_{ia}(t) \equiv S_i(a + n_t)$ for $a \leq t\%D$ and $S_{ia}(t) \equiv S_i(a + n_t - D)$ for $a > t\%D$, where $n_t$ is defined through $t \equiv n_t + t\%D$. Due to (6), *symmetric* efficacies $J_{ij}^{ab} = J_{ji}^{ba}$ may be contructed for the new system by

$$J_{ij}^{ab} = J_{ij}((b-a-1)\%D) , \qquad (9)$$

allowing a well-defined Hamiltonian, equal to that of a Hopfield net of size $ND$,

$$H = -\frac{1}{2} \sum_{i,j=1}^{N} \sum_{a,b=0}^{D-1} J_{ij}^{ab} S_{ia} S_{jb} . \qquad (10)$$

An evaluation of (10) in terms of the former state variables reveals that it is identical to the Lyapunov functional (7). The interpretation, however, is changed: a *limit cycle* of period $D$ in the original network corresponds to a *fixed-point* of the new system of size $ND$. We have thus shown that the time evolution of a delay network with extended symmetry can be understood in terms of a downhill motion in the energy landscape of its "equivalent system".

For Hebbian couplings (5), the new efficacies $J_{ij}^{ab}$ take a particularly simple form if we define patterns $\{\xi_{ia}^{\mu\alpha}; 1 \leq i \leq N, 0 \leq \alpha \leq D{-}1\}$ by $\xi_{ia}^{\mu\alpha} \equiv \xi_{i,(a-\alpha)\%D}^{\mu 0}$, i.e., if we create column-shifted copies of the prototype $\xi_{ia}^{\mu 0}$. Setting $\mathcal{E}_{ab} \equiv \varepsilon((b-a-1)\%D) = \mathcal{E}_{ba}$ leads to

$$J_{ij}^{ab} = \mathcal{E}_{ab} N^{-1} \sum_{\mu=1}^{P} \sum_{\alpha=0}^{D-1} \xi_{ia}^{\mu\alpha} \xi_{jb}^{\mu\alpha} \ . \tag{11}$$

Storing one cycle $\sigma_i(t_\mu) = \xi_{ia}^{\mu 0}$ in the delay network thus corresponds to memorizing $D$ shifted duplicates $\xi_{ia}^{\mu\alpha}$, $0 \leq \alpha < D$, in the equivalent system, reflecting that a $D$-cycle can be retrieved in $D$ different time-shifted versions in the original network.

If, in the second step, we now switch to the stochastic dynamics (2), the important question arises whether $H$ also determines the equilibrium distribution $\rho$ of the system. This need not be true since the column-wise dynamics of the equivalent network differs from both the Little and Hopfield model. An elaborate proof (Li and Herz, 1990), however, shows that there is indeed an equilibrium distribution *à la* Gibbs,

$$\rho(A) = Z^{-1} \exp[-\beta H(A)] \ , \tag{12}$$

where $Z \equiv \mathrm{Tr}_\mathbf{A} \exp[-\beta H(A)]$. In passing we note that for $D = 2$ there are only links with zero delay. By (6) we have $J_{ij}(0) = J_{ji}(0)$, i.e., we are dealing with a symmetric Little model. We may introduce a reduced probability distribution $\tilde{\rho}$ for this special case, $\tilde{\rho}(A_1) \equiv \mathrm{Tr}_{\mathbf{A}_0} \rho(A_0 A_1)$, and obtain $\tilde{\rho}(A_1) = Z^{-1} \exp[-\beta \tilde{H}(A_1)]$ with

$$\tilde{H} \equiv -\beta^{-1} \sum_{i=1}^{N} \ln[2\cosh(\beta \sum_{j=1}^{N} J_{ij} S_j)] \ . \tag{13}$$

We thus have recovered both the effective Hamiltonian of the Little model as derived by Peretto (1984) and the duplicated-system technique of van Hemmen (1986).

We finish our argument by turning to quantative results. We focus on the case where each of the $P$ learning sessions corresponds to teaching a (different) cycle of $D$ patterns $\xi_{ia}^{\mu 0}$, each lasting for one time step. We work with unbiased random patterns where $\xi_{ia}^{\mu 0} = \pm 1$ with equal probability, and study our network at a finite storage level $\alpha = \lim_{N\to\infty}(P/N) > 0$. A detailed analysis of the case where the number of cycles remains bounded as $N \to \infty$ can be found in (Li and Herz 1990).

As in the replica-symmetric theory of Amit et al. (1987), we assume that the network is in a state highly correlated with a *finite* number of stored cycles. The remaining, extensively many cycles are described as a noise term. We define "partial" overlaps by $m_a^{\mu\alpha} \equiv N^{-1} \sum_i \xi_{ia}^{\mu\alpha} S_{ia}$. These macroscopic order parameters measure how close the system is to a stored pattern $\xi^{\mu\alpha}$ at a specific column $a$. We consider *retrieval*

*solutions*, i.e., $m_a^{\mu\alpha} = m^\mu \delta_{\alpha,0}$, and arrive at the fixed-point equations (Li and Herz 1990)

$$m^\mu = \left\langle\!\left\langle \xi^{\mu 0} \tanh[\beta\{\sum_\nu m^\nu \xi^{\nu 0} + \sqrt{\alpha r} z\}]\right\rangle\!\right\rangle , \qquad (14)$$

where

$$r = q \sum_{k=1}^{D} \frac{[\lambda_k(\mathcal{E})]^2}{[1 - \beta(1-q)\lambda_k(\mathcal{E})]^2} \quad \text{and} \quad q = \left\langle\!\left\langle \tanh^2[\beta\{\sum_\nu m^\nu \xi^{\nu 0} + \sqrt{\alpha r} z\}]\right\rangle\!\right\rangle. \quad (15)$$

Double angular brackets represent an average with respect to both the "condensed" cycles and the normalized Gaussian random variable $z$. The $\lambda_k(\mathcal{E})$ are eigenvalues of the matrix $\mathcal{E}$. Retrieval is possible when solutions with $m^\mu > 0$ for a single cycle $\mu$ exist, and the storage capacity $\alpha_c$ is reached when such solutions cease to exist. It should be noted that each cycle consists of $D$ patterns so that the storage capacity for *single* patterns is $\tilde{\alpha}_c = D\alpha_c$. During the recognition process, however, each of them will trigger the cycle it belongs to and cannot be retrieved as a static pattern. For systems with a "maximally uniform" distribution, $\mathcal{E}_{ab} = (D-1)^{-1}(1-\delta_{ab})$, we get

| D | 2 | 3 | 4 | 5 | $\infty$ |
|---|---|---|---|---|---|
| $\alpha_c$ | 0.100 | 0.110 | 0.116 | 0.120 | 0.138 |

where the last result is identical to that for the corresponding Hopfield model since the diagonal terms of $\mathcal{E}$ can be neglected in that case. The above findings agree well with estimates from a finite-size analysis ($N \leq 3000$) of data from numerical simulations as shown by two examples. For $D=3$, we have found $\alpha_c = 0.120 \pm 0.015$, for $D=4$, $\alpha_c = 0.125 \pm 0.015$. Our results demonstrate that the storage capacity for temporal associations is comparable to that for static memories. As an example, take $D=2$, i.e., the Little model. In the limit of large $N$, we see that $0.100 \cdot N$ two-cycles of the form $\xi_{i0}^{\mu 0} \rightleftharpoons \xi_{i1}^{\mu 0}$ may be recalled as compared to $0.138 \cdot N$ static patterns (Fontanari and Köberle 1987); this leads to an 1.45-fold increase of the information content per synapse.

The influence of the weight distribution on the network behavior may be demonstrated by some choices of $\varepsilon(\tau)$ for $D=4$:

| $\tau$ = | 0 | 1 | 2 | 3 | $\alpha_c$ | $m_c$ |
|---|---|---|---|---|---|---|
| $\varepsilon(\tau)$ = | 1/3 | 1/3 | 1/3 | 0 | 0.116 | 0.96 |
| $\varepsilon(\tau)$ = | 1/2 | 0 | 1/2 | 0 | 0.100 | 0.93 |
| $\varepsilon(\tau)$ = | 0 | 1 | 0 | 0 | 0.050 | 0.93 |

The storage capacity decreases with decreasing number of delay lines, but measured *per synapse*, it does increase. However, networks with only a few number of delays are less fault-tolerant as known from numerical simulations (Herz et al. 1989). For all studied architectures, retrieved sequences contain less than 3.5% errors.

Our results prove that an extensive number of temporal associations can be stored as spatio-temporal attractors for the retrieval dynamics. They also indicate that dynamical systems with delayed interactions can be programmed in a very efficient manner to perform associative computations in the space-time domain.

## 6 CONCLUSION

Learning schemes can be successful only if the structure of the learning task is compatible with both the network architecture and the learning algorithm. In the present context, the task is to store simple temporal associations. It can be accomplished in neural networks with a broad distribution of signal delays and Hebbian synapses which, during learning periods, operate as microscopic feature detectors for spatio-temporal correlations within the external stimuli. The retrieval dynamics utilizes the very same delays and synapses, and is therefore rather robust as shown by numerical simulations and a statistical mechanical analysis.

Our approach may be generalized in various directions. For example, one can investigate more sophisticated learning rules or switch to continuous neurons in "iterated-map networks" (Marcus and Westervelt 1990). A generalization of the Lyapunov functional (7) covers that case as well (Herz, to be published) and allows a direct comparison of theoretical predictions with results from hardware implementations. Finally, one could try to develop a Lyapunov functional for a continuous-time dynamics with delays which seems to be rather significant for applications as well as for the general theory of functional differential equations and dynamical systems.

### Acknowledgements

It is a pleasure to thank Bernhard Sulzer, John Hopfield, Reimer Kühn and Wulfram Gerstner for many helpful discussions. AVMH acknowledges support from the Studienstiftung des Deutschen Volkes. ZL is partly supported by a grant from the Seaver Institute.

### References

Amit D J, Gutfreund H and Sompolinsky H 1985 *Phys. Rev.* **A 32** 1007
— 1987 *Ann. Phys. (N.Y.)* **173** 30
Coolen A C C and Gielen C C A M 1988 *Europhys. Lett.* **7** 281
Fontanari J F and Köberle R 1987 *Phys. Rev.* **A 36** 2475
Hebb D O 1949 *The Organization of Behavior* Wiley, New York
van Hemmen J L 1986 *Phys. Rev.* **A 34** 3435
Herz A V M, Sulzer B, Kühn R and van Hemmen J L 1988 *Europhys. Lett.* **7** 663
— 1989 *Biol. Cybern.* 60 457
Hopfield J J 1982 *Proc. Natl. Acad. Sci. USA* **79** 2554
Kerszberg M and Zippelius A 1990 *Phys. Scr.* **T33** 54
Kleinfeld D 1986 *Proc. Natl. Acad. Sci. USA* **83** 9469
Li Z and Herz A V M 1990 in *Lecture Notes in Physics* **368** pp287 Springer, Heidelberg
Little W A 1974 *Math. Biosci.* **19** 101
Marcus C M and Westervelt R M 1990 *Phys. Rev.* **A 42** 2410
Peretto P 1984 *Biol. Cybern.* **50** 51
Sompolinsky H and Kanter I 1986 *Phys. Rev. Lett.* **57** 2861
Tank D W and Hopfield J J 1987 *Proc. Natl. Acad. Sci. USA* **84** 1896
